# Generative Local Metric Learning for Nearest Neighbor Classification

**Yung-Kyun Noh**[1,2]    **Byoung-Tak Zhang**[2]    **Daniel D. Lee**[1]
[1]GRASP Lab, University of Pennsylvania, Philadelphia, PA 19104, USA
[2]Biointelligence Lab, Seoul National University, Seoul 151-742, Korea
nohyung@seas.upenn.edu, btzhang@snu.ac.kr, ddlee@seas.upenn.edu

## Abstract

We consider the problem of learning a local metric to enhance the performance of nearest neighbor classification. Conventional metric learning methods attempt to separate data distributions in a purely discriminative manner; here we show how to take advantage of information from parametric generative models. We focus on the bias in the information-theoretic error arising from finite sampling effects, and find an appropriate local metric that maximally reduces the bias based upon knowledge from generative models. As a byproduct, the asymptotic theoretical analysis in this work relates metric learning with dimensionality reduction, which was not understood from previous discriminative approaches. Empirical experiments show that this learned local metric enhances the discriminative nearest neighbor performance on various datasets using simple class conditional generative models.

## 1  Introduction

The classic dichotomy between generative and discriminative methods for classification in machine learning can be clearly seen in two distinct performance regimes as the number of training examples is varied [12, 18]. Generative models—which employ models first to find the underlying distribution $p(\mathbf{x}|y)$ for discrete class label $y$ and input data $\mathbf{x} \in \mathbb{R}^D$—typically outperform discriminative methods when the number of training examples is small, due to smaller variance in the generative models which compensates for any possible bias in the models. On the other hand, more flexible discriminative methods—which are interested in a direct measure of $p(y|\mathbf{x})$—can accurately capture the true posterior structure $p(y|\mathbf{x})$ when the number of training examples is large. Thus, given enough training examples, the best performing classification algorithms have typically employed purely discriminative methods.

However, due to the curse of dimensionality when $D$ is large, the number of data examples may not be sufficient for discriminative methods to approach their asymptotic performance limits. In this case, it may be possible to improve discriminative methods by exploiting knowledge of generative models. There has been recent work on hybrid models showing some improvement [14, 15, 20], but mainly the generative models have been improved through the discriminative formulation. In this work, we consider a very simple discriminative classifier, the nearest neighbor classifier, where the class label of an unknown datum is chosen according to the class label of the *nearest* known datum. The choice of a metric to define *nearest* is then crucial, and we show how this metric can be locally defined based upon knowledge of generative models.

Previous work on metric learning for nearest neighbor classification has focused on a purely discriminative approach. The metric is parameterized by a global quadratic form which is then optimized on the training data to maximize pairwise separation between dissimilar points, and to minimize the pairwise separation of similar points [3, 9, 10, 21, 26]. Here, we show how the problem of learning

a metric can be related to reducing the theoretical bias of the nearest neighbor classifier. Though the performance of the nearest neighbor classifier has good theoretical guarantees in the limit of infinite data, finite sampling effects can introduce a bias which can be minimized by the choice of an appropriate metric. By directly trying to reduce this bias at each point, we will see the classification error is significantly reduced compared to the global class-separating metric.

We show how to choose such a metric by analyzing the probability distribution on nearest neighbors, provided we know the underlying generative models. Analyses of nearest neighbor distributions have been discussed before [11, 19, 24, 25], but we take a simpler approach and derive the metric-dependent term in the bias directly. We then show that minimizing this bias results in a semi-definite programming optimization that can be solved analytically, resulting in a locally optimal metric. In related work, Fukunaga et al. considered optimizing a metric function in a generative setting [7, 8], but the resulting derivation was inaccurate and does not improve nearest neighbor performance. Jaakkola et al. first showed how a generative model can be used to derive a special kernel, called the Fisher kernel [12], which can be related to a distance function. Unfortunately, the Fisher kernel is quite generic, and need not necessarily improve nearest neighbor performance.

Our generative approach also provides a theoretical relationship between metric learning and the dimensionality reduction problem. In order to find better projections for classification, research on dimensionality reduction using labeled training data has utilized information-theoretic measures such as Bhattacharrya divergence [6] and mutual information [2, 17]. We argue how these problems can be connected with metric learning for nearest neighbor classification within the general framework of F-divergences. We will also explain how dimensionality reduction is entirely different from metric learning in the generative approach, whereas in the discriminative setting, it is simply a special case of metric learning where particular directions are shrunk to zero.

The remainder of the paper is organized as follows. In section 2, we motivate by comparing the metric dependency of the discriminative and generative approaches for nearest neighbor classification. After we derive the bias due to finite sampling in section 3, we show, in section 4, how minimizing this bias results in a local metric learning algorithm. In section 5, we explain how metric learning should be understood in a generative perspective, in particular, its relationship with dimensionality reduction. Experiments on various datasets are presented in section 6, comparing our experimental results with other well-known algorithms. Finally, in section 7, we conclude with a discussion of future work and possible extensions.

## 2 Metric and Nearest Neighbor Classification

In recent work, determining a good metric for nearest neighbor classification is believed to be crucial. However, traditional generative analysis of this problem has simply ignored the metric issue with good reason, as we will see in section 2.2. In this section, we explain the apparent contradiction between two different approaches to this issue, and briefly describe how the resolution of this contradiction will lead to a metric learning method that is both theoretically and practically plausible.

### 2.1 Metric Learning for Nearest Neighbor Classification

A nearest neighbor classifier determines the label of an unknown datum according to the label of its nearest neighbor. In general, the meaning of the term *nearest* is defined along with the notion of distance in data space. One common choice for this distance is the Mahalanobis distance with a positive definite square matrix $A \in \mathbb{R}^{D \times D}$ where $D$ is the dimensionality of data space. In this case, the distance between two points $\mathbf{x}_1$ and $\mathbf{x}_2$ is defined as

$$d(\mathbf{x}_1, \mathbf{x}_2) = \sqrt{(\mathbf{x}_1 - \mathbf{x}_2)^T A (\mathbf{x}_1 - \mathbf{x}_2)} \,, \tag{1}$$

and the nearest datum $\mathbf{x}_{NN}$ is one having minimal distance to the test point among labeled training data in $\{\mathbf{x}_i\}_{i=1}^N$

In this classification task, the results are highly dependent on the choice of matrix $A$, and prior work has attempted to improve the performance by a better choice of $A$. This recent work has assumed the following common heuristic: the training data in different classes should be separated in a new

metric space. Given training data, a global $A$ is optimized such that directions separating different class data are extended, and directions binding same class data together are shrunk [3, 9, 10, 21, 26].

However, in terms of the test results, these conventional methods do not improve the performance dramatically, which will be shown in our later experiments on large datasets, and we show why only small improvements arise in our theoretical analysis.

## 2.2 Theoretical Performance of Nearest Neighbor Classifier

Contrary to recent metric learning approaches, a simple theoretical analysis using a generative model displays no sensitivity to the choice of the metric. We consider i.i.d. samples generated from two different distributions $p_1(\mathbf{x})$ and $p_2(\mathbf{x})$ over the vector space $\mathbf{x} \in \mathbb{R}^D$. With infinite samples, the probability of misclassification using a nearest neighbor classifier can be obtained:

$$E_{Asymp} = \int \frac{p_1(\mathbf{x})p_2(\mathbf{x})}{p_1(\mathbf{x}) + p_2(\mathbf{x})} d\mathbf{x}, \tag{2}$$

which is better known by its relationship to an upper bound, twice the optimal Bayes error [4, 7, 8].

By looking at the asymptotic error in a linearly transformed $\mathbf{z}$-space, we can show that Eq. (2) is invariant to the change of metric. If we consider a linear transformation $\mathbf{z} = L^T \mathbf{x}$ using a full rank matrix $L$, and the distribution $q_c(\mathbf{z})$ for $c \in \{1, 2\}$ in $\mathbf{z}$-space satisfying $p_c(\mathbf{x})d\mathbf{x} = q_c(\mathbf{z})d\mathbf{z}$ and accompanying measure change $d\mathbf{z} = |L|d\mathbf{x}$, we see $E_{Asymp}$ in $\mathbf{z}$-space is unchanged. Since any positive definite $A$ can be decomposed as $A = LL^T$, we can say the asymptotic error remains constant even as the metric shrinks or expands any spatial directions in data space.

This difference in behavior in terms of metric dependence can be understood as a special property that arises from infinite data. When we do not have infinite samples, the expectation of error is biased in that it deviates from the asymptotic error, and the bias is dependent on the metric. From a theoretical perspective, the asymptotic error is the theoretical limit of expected error, and the bias reduces as the number of samples increase. Since this difference is not considered in previous research, the aforementioned metric will not exhibit performance improvements when the sample number is large.

In the next section, we look at the performance bias associated with finite sampling directly and find a metric that minimizes the bias from the asymptotic theoretical error.

## 3 Performance Bias due to Finite Sampling

Here, we obtain the expectation of nearest neighbor classification error from the distribution of nearest neighbors in different classes. As we consider finite number of samples, the nearest neighbor from a point $\mathbf{x}_0$ appears at a finite distance $d_N > 0$. This non-zero distance gives rise to the performance difference from its theoretical limit (2). A twice-differentiable distribution $p(\mathbf{x})$ is considered and approximated to second order near a test point $\mathbf{x}_0$:

$$p(\mathbf{x}) \simeq p(\mathbf{x}_0) + \nabla p(\mathbf{x})|^T_{\mathbf{x}=\mathbf{x}_0}(\mathbf{x} - \mathbf{x}_0) + \frac{1}{2}(\mathbf{x} - \mathbf{x}_0)^T \nabla \nabla p(\mathbf{x})\big|_{\mathbf{x}=\mathbf{x}_0}(\mathbf{x} - \mathbf{x}_0) \tag{3}$$

with the gradient $\nabla p(\mathbf{x})$ and Hessian matrix $\nabla \nabla p(\mathbf{x})$ defined by taking derivatives with respect to $\mathbf{x}$.

Now, under the condition that the nearest neighbor appears at the distance $d_N$ from the test point, the expectation of the probability $p(\mathbf{x}_{NN})$ at a nearest neighbor point is derived by averaging the probability over the $D$-dimensional hypersphere of radius $d_N$, as in Fig. 1. After averaging, the gradient term disappears, and the resulting expectation is the sum of the probability at $\mathbf{x}_0$ and a residual term containing the Laplacian of $p$. We replace this expected probability by $\tilde{p}(\mathbf{x}_0)$.

$$\begin{aligned}
&E_{\mathbf{x}_{NN}}\Big[p(\mathbf{x}_{NN})\Big|d_N, \mathbf{x}_0\Big] \\
=\ & p(\mathbf{x}_0) + \frac{1}{2}E_{\mathbf{x}_{NN}}\Big[(\mathbf{x} - \mathbf{x}_0)^T \nabla \nabla p(\mathbf{x})(\mathbf{x} - \mathbf{x}_0)\Big|\|\mathbf{x} - \mathbf{x}_0\|^2 = d_N^2\Big] \\
=\ & p(\mathbf{x}_0) + \frac{d_N^2}{2D} \cdot \nabla^2 p|_{\mathbf{x}=\mathbf{x}_0} \equiv \tilde{p}(\mathbf{x}_0)
\end{aligned} \tag{4}$$

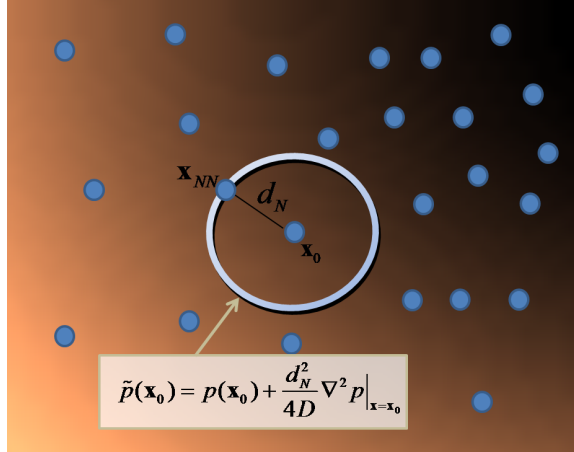

Figure 1: The nearest neighbor $\mathbf{x}_{NN}$ appears at a finite distance $d_N$ from $\mathbf{x}_0$ due to finite sampling. Given the data distribution $p(\mathbf{x})$, the average probability density function over the surface of a $D$ dimensional hypersphere is $\tilde{p}(\mathbf{x}_0) = p(\mathbf{x}_0) + \frac{d_N^2}{4D}\nabla^2 p|_{\mathbf{x}=\mathbf{x}_0}$ for small $d_N$.

where the scalar Laplacian $\nabla^2 p(\mathbf{x})$ is given by the sum of the eigenvalues of the Hessian $\nabla\nabla p(\mathbf{x})$.

If we look at the expected error, it is the expectation of the probability that the test point and its neighbor are labeled differently. In other words, the expectation error $E_{NN}$ is the expectation of $e(\mathbf{x}, \mathbf{x}_{NN}) = p(C_1|\mathbf{x})p(C_2|\mathbf{x}_{NN}) + p(C_2|\mathbf{x})p(C_1|\mathbf{x}_{NN})$ over both the distribution of $\mathbf{x}$ and the distribution of nearest neighbor $\mathbf{x}_{NN}$ for a given $\mathbf{x}$:

$$E_{NN} = E_{\mathbf{x}}\left[E_{\mathbf{x}_{NN}}\left[e(\mathbf{x}, \mathbf{x}_{NN})\Big|\mathbf{x}\right]\right] \tag{5}$$

We then replace the posteriors $p(C|\mathbf{x})$ and $p(C|\mathbf{x}_{NN})$ as $p_c(\mathbf{x})/(p_1(\mathbf{x}) + p_2(\mathbf{x}))$ and $p_c(\mathbf{x}_{NN})/(p_1(\mathbf{x}_{NN}) + p_2(\mathbf{x}_{NN}))$ respectively, and approximate the expectation of the posterior $E_{\mathbf{x}_{NN}}\left[p(C|\mathbf{x}_{NN})\Big|d_N, \mathbf{x}\right]$ at a fixed distance $d_N$ from test point $\mathbf{x}$ using $\tilde{p}_c(\mathbf{x})/(\tilde{p}_1(\mathbf{x}) + \tilde{p}_2(\mathbf{x}))$. If we expand $E_{NN}$ with respect to $d_N$, and take the expectation using the decomposition, $E_{\mathbf{x}_{NN}}\left[f\right] = E_{d_N}\left[E_{\mathbf{x}_{NN}}\left[f\Big|d_N\right]\right]$, then the expected error is given to leading order by

$$
\begin{aligned}
E_{NN} \quad &\simeq \quad \int \frac{p_1 p_2}{p_1 + p_2}d\mathbf{x} + \\
&\frac{E_{d_N}[d_N^2]}{4D}\int \frac{1}{(p_1 + p_2)^2}\left[p_1^2\nabla^2 p_2 + p_2^2\nabla^2 p_1 - p_1 p_2(\nabla^2 p_1 + \nabla^2 p_2)\right]d\mathbf{x}
\end{aligned}
\tag{6}
$$

When $E_{d_N}[d_N^2] \to 0$ with an infinite number of samples, this error converges to the asymptotic limit in Eq. (2) as expected. The residual term can be considered as the finite sampling bias of the error discussed earlier. Under the coordinate transformation $\mathbf{z} = L^T\mathbf{x}$ and the distributions $p(\mathbf{x})$ on $\mathbf{x}$ and $q(\mathbf{z})$ on $\mathbf{z}$, we see that this bias term is dependent upon the choice of a metric $A = LL^T$.

$$
\frac{1}{(q_1 + q_2)^2}\left[q_1^2\nabla^2 q_2 + q_2^2\nabla^2 q_1 - q_1 q_2\left(\nabla^2 q_1 + \nabla^2 q_2\right)\right]d\mathbf{z}
\tag{7}
$$

$$
= \frac{1}{(p_1 + p_2)^2}tr\left[A^{-1}\left(p_1^2\nabla\nabla p_2 + p_2^2\nabla\nabla p_1 - p_1 p_2\left(\nabla\nabla p_1 + \nabla\nabla p_2\right)\right)\right]d\mathbf{x}
$$

which is derived using $p(\mathbf{x})d\mathbf{x} = q(\mathbf{z})d\mathbf{z}$ and $|L|\nabla^2 q = tr[A^{-1}\nabla\nabla p]$. Expectation of squared distance $E_{d_N}[d_N^2]$ is related to the determinant $|A|$, which will be fixed to 1. Thus, finding the metric that minimizes the quantity given in Eq. (7) at each point is equivalent to minimizing the metric-dependent bias in Eq. (6).

## 4 Reducing Deviation from the Asymptotic Performance

Finding the local metric that minimizes the bias can be formulated as a semi-definite programming (SDP) problem of minimizing squared residual with respect to a positive semi-definite metric $A$:

$$\min_{A} (tr[A^{-1}B])^2 \quad s.t. \quad |A| = 1, A \succeq 0 \tag{8}$$

where the matrix $B$ at each point is

$$B = p_1^2 \nabla\nabla p_2 + p_2^2 \nabla\nabla p_1 - p_1 p_2 (\nabla\nabla p_1 + \nabla\nabla p_2). \tag{9}$$

This is a simple SDP having an analytical solution where the solution shares the eigenvectors with $B$. Let $\Lambda_+ \in \mathbb{R}^{d_+ \times d_+}$ and $\Lambda_- \in \mathbb{R}^{d_- \times d_-}$ be the diagonal matrices containing the positive and negative eigenvalues of $B$ respectively. If $U_+ \in \mathbb{R}^{D \times d_+}$ contains the eigenvectors corresponding to the eigenvalues in $\Lambda_+$ and $U_- \in \mathbb{R}^{D \times d_-}$ contains the eigenvectors corresponding to the eigenvalues in $\Lambda_-$, we use the solution given by

$$A_{opt} = [U_+ \ U_-] \begin{pmatrix} d_+ \Lambda_+ & 0 \\ 0 & -d_- \Lambda_- \end{pmatrix} [U_+ \ U_-]^T \tag{10}$$

The solution $A_{opt}$ is a local metric since we assumed that the nearest neighbor was close to the test point satisfying Eq. (3). In principle, distances should then be defined as geodesic distances using this local metric on a Riemannian manifold. However, this is computationally difficult, so we use the surrogate distance $A = \gamma I + A_{opt}$ and treat $\gamma$ as a regularization parameter that is learned in addition to the local metric $A_{opt}$.

The multiway extension of this problem is straightforward. The asymptotic error with $C$-class distributions can be extended to $\frac{1}{C} \sum_{c=1}^{C} \int \left( p_c \sum_{j \neq i} p_j \right) / \left( \sum_i p_i \right) d\mathbf{x}$ using the posteriors of each class, and it replaces $B$ in Eq. (9) by the extended matrix:

$$B = \sum_{i=1}^{C} \nabla^2 p_i \left( \sum_{j \neq i} p_j^2 - p_i \sum_{j \neq i} p_j \right). \tag{11}$$

## 5 Metric Learning in Generative Models

Traditional metric learning methods can be understood as being purely discriminative. In contrast to our method that directly considers the expected error, those methods are focused on maximizing the separation of data belonging to different classes. In general, their motivations are compared to the supervised dimensionality reduction methods, which try to find a low dimensional space where the separation between classes is maximized. Their dimensionality reduction is not that different from metric learning, but often as a special case where metric in particular directions is forced to be zero.

In the generative approach, however, the relationship between dimensionality reduction and metric learning is different. As in the discriminative case, dimensionality reduction in generative models tries to obtain class separation in a transformed space. It assumes particular parametric distributions (typically Gaussians), and uses a criterion to maximize the separation [2, 6, 16, 17]. One general form of these criteria is the F-divergence (also known as Csiszer's general measure of divergence), that can be defined with respect to a convex function $\phi(t)$ for $t \in \mathbb{R}$ [13]:

$$F(p_1(\mathbf{x}), p_2(\mathbf{x})) = \int p_1(\mathbf{x}) \, \phi\left( \frac{p_2(\mathbf{x})}{p_1(\mathbf{x})} \right) d\mathbf{x}. \tag{12}$$

The examples of using this divergence include the Bhattacharyya divergence $\int \sqrt{p_1(\mathbf{x})p_2(\mathbf{x})} d\mathbf{x}$ when $\phi(t) = \sqrt{t}$ and the KL-divergence $-\int p_1(\mathbf{x}) \log\left( \frac{p_2(\mathbf{x})}{p_1(\mathbf{x})} \right) d\mathbf{x}$ when $\phi(t) = -\log(t)$. Using mutual information between data and labels can be understood as an extension of KL-divergence. The well known Linear Discriminant Analysis is a special example of Bhattacharyya criterion when we assume two-class Gaussians sharing the same covariance matrices.

Unlike dimensionality reduction, we cannot use these criteria for metric learning because any F-divergence is metric-invariant. The asymptotic error Eq. (2) is related to one particular F-divergence

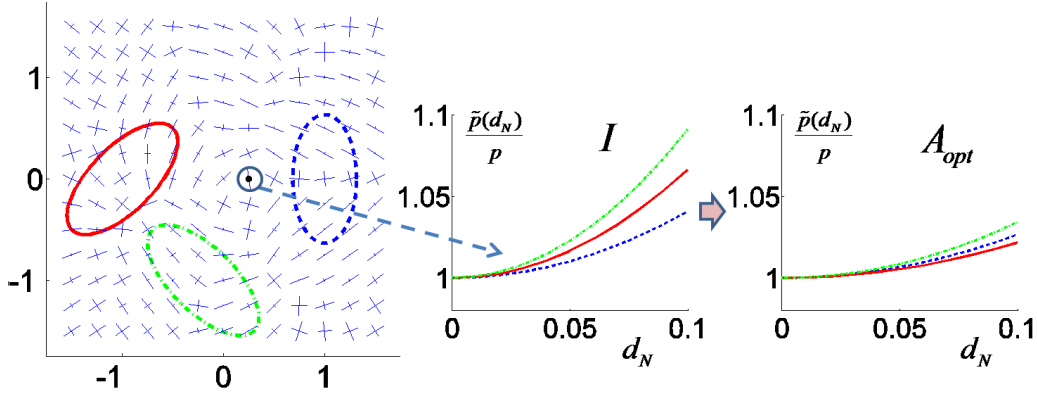

Figure 2: Optimal local metrics are shown on the left for three example Gaussian distributions in a 5-dimensional space. The projected 2-dimensional distributions are represented as ellipses (one standard deviation from the mean), while the remaining 3 dimensions have an isotropic distribution. The local $\tilde{p}/p$ of the three classes are plotted on the right using a Euclidean metric $I$ and for the optimal metric $A_{opt}$. The solution $A_{opt}$ tries to keep the ratio $\tilde{p}/p$ over the different classes as similar as possible when the distance $d_N$ is varied.

by $E_{Asymp} = 1 - F(p_1, p_2)$ with a convex function $\phi(t) = 1/(1 + t)$. Therefore, in generative models, the metric learning problem is qualitatively different from the dimensionality reduction problem in this aspect. One interpretation is that the F-measure can be understood as a measure of dimensionality reduction in an asymptotic situation. In this case, the role of metric learning can be defined to move the expected F-measure toward the asymptotic F-measure by appropriate metric adaptation.

Finally, we provide an alternative understanding on the problem of reducing Eq. (7). By reformulating Eq. (9) into $(p_2 - p_1)(p_2 \nabla^2 p_1 - p_1 \nabla^2 p_2)$, we can see that the optimal metric tries to minimize the difference between $\frac{\nabla^2 p_1}{p_1}$ and $\frac{\nabla^2 p_2}{p_2}$. If $\frac{\nabla^2 p_1}{p_1} \approx \frac{\nabla^2 p_2}{p_2}$, this also implies

$$\frac{\tilde{p}_1}{p_1} \approx \frac{\tilde{p}_2}{p_2} \tag{13}$$

for $\tilde{p} = p + \frac{d_N^2}{2D} \nabla^2 p$, the average probability at a distance $d_N$ in (4). Thus, the algorithm tries to keep the ratio of the average probabilities $\tilde{p_1}/\tilde{p_2}$ at a distance $d_N$ to be as similar as possible to the ratio of probabilities $p_1/p_2$ at the test point. This means that the expected nearest neighbor classification at a distance $d_N$ will be least biased due to finite sampling. Fig. 2 shows how the learned local metric $A_{opt}$ varies at a point $\mathbf{x}$ for a 3-class Gaussian example, and how the ratio of $\tilde{p}/p$ is kept as similar as possible.

## 6   Experiments

We apply our algorithm for learning a local metric to synthetic and various real datasets and see how well it improves nearest neighbor classification performance. Simple standard Gaussian distributions are used to learn the generative model, with parameters including the mean vector $\mu$ and covariance matrix $\Sigma$ for each class. The Hessian of a Gaussian distribution is then given by the expression:

$$\nabla \nabla p(\mathbf{x}) = p(\mathbf{x}) \Big[ \Sigma^{-1} (\mathbf{x} - \mu)(\mathbf{x} - \mu)^T \Sigma^{-1} - \Sigma^{-1} \Big] \tag{14}$$

This expression is then used to learn the optimal local metric. We compare the performance of our method (GLML—Generative Local Metric Learning) with recent metric learning discriminative methods which report state-of-the-art performance on a number of datasets. These include

Information-Theoretic Metric Learning (ITML)[1] [3], Boost Metric[2] (BM) [21], and Largest Margin Nearest Neighbor (LMNN)[3] [26]. We used the implementations downloaded from the corresponding authors' websites. We also compare with a local metric given by the Fisher kernel [12] assuming a single Gaussian for the generative model and using the location parameter to derive the Fisher information matrix. The metric from the Fisher kernel was not originally intended for nearest neighbor classification, but it is the only other reported algorithm that learns a local metric from generative models.

For the synthetic dataset, we generated data from two-class random Gaussian distributions having two fixed means. The covariance matrices are generated from random orthogonal eigenvectors and random eigenvalues. Experiments were performed varying the input dimensionality, and the classification accuracies are shown in Fig. 3.(a) along with the results of the other algorithms. We used 500 test points and an equal number of training examples. The experiments were performed with 20 different realizations and the results were averaged. As the dimensionality grows, the original nearest neighbor performance degrades because of the high dimensionality. However, we see that the proposed local metric highly outperforms the discriminative nearest neighbor performance in a high dimensional space appropriately. We note that this example is ideal for GLML, and it shows much improvement compared to the other methods.

The other experiments consist of the following benchmark datasets: UCI machine learning repository[4] datasets (Ionosphere, Wine), and the IDA benchmark repository[5] (German, Image, Waveform, Twonorm). We also used the USPS handwritten digits and the TI46 speech dataset. For the USPS data, we resized the images to $8 \times 8$ pixels and trained on the 64-dimensional pixel vector data. For the TI46 dataset, the examples consist of spoken sounds pronounced by 8 different men and 8 different women. We chose the pronunciation of ten digits ("zero" to "nine"), and performed a 10 class digit classification task. 10 different filters in the Fourier domain were used as features to preprocess the acoustic data. The experiments were done on 20 data sampling realizations for Twonorm and TI46, 10 for USPS, 200 for Wine, and 100 for the others.

Except the synthetic data in Fig. 3.(a), the data consist of various number of training data per class. The regularization parameter $\gamma$ value is chosen by cross-regularization on a subset of the training data, then fixed for testing. The covariance matrix of the learned Gaussian distributions is also regularized by setting $\Sigma = \hat{\Sigma} + \alpha I$ where $\hat{\Sigma}$ is the estimated covariance. The parameter $\alpha$ is set prior to each experiment.

From the results shown in Fig. 3, our local metric algorithm generally outperforms most of the other metrics across most of the datasets. On quite a number of datasets, many of the other methods do not outperform the original Euclidean nearest neighbor classifier. This is because on some of these datasets, performance cannot be improved using a global metric. On the other hand, the local metric derived from simple Gaussian distributions always shows a performance gain over the naive nearest neighbor classifier. In contrast, using Bayes rule with these simple Gaussian generative models often results in very poor performance. The computational time using a local metric is also very competitive, since the underlying SDP optimization has a simple spectral solution. This is in contrast to other methods which numerically solve for a global metric using an SDP over the data points.

# 7 Conclusions

In our study, we showed how a local metric for nearest neighbor classification can be learned using generative models. Our experiments show improvement over competitive methods on a number of experimental datasets. The learning algorithm is derived from an analysis of the asymptotic performance of the nearest neighbor classifier, such that the optimal metric minimizes the bias of the expected performance of the classifier. This connection to generative models is very powerful, and can easily be extended to include missing data—one of the large advantages of generative models

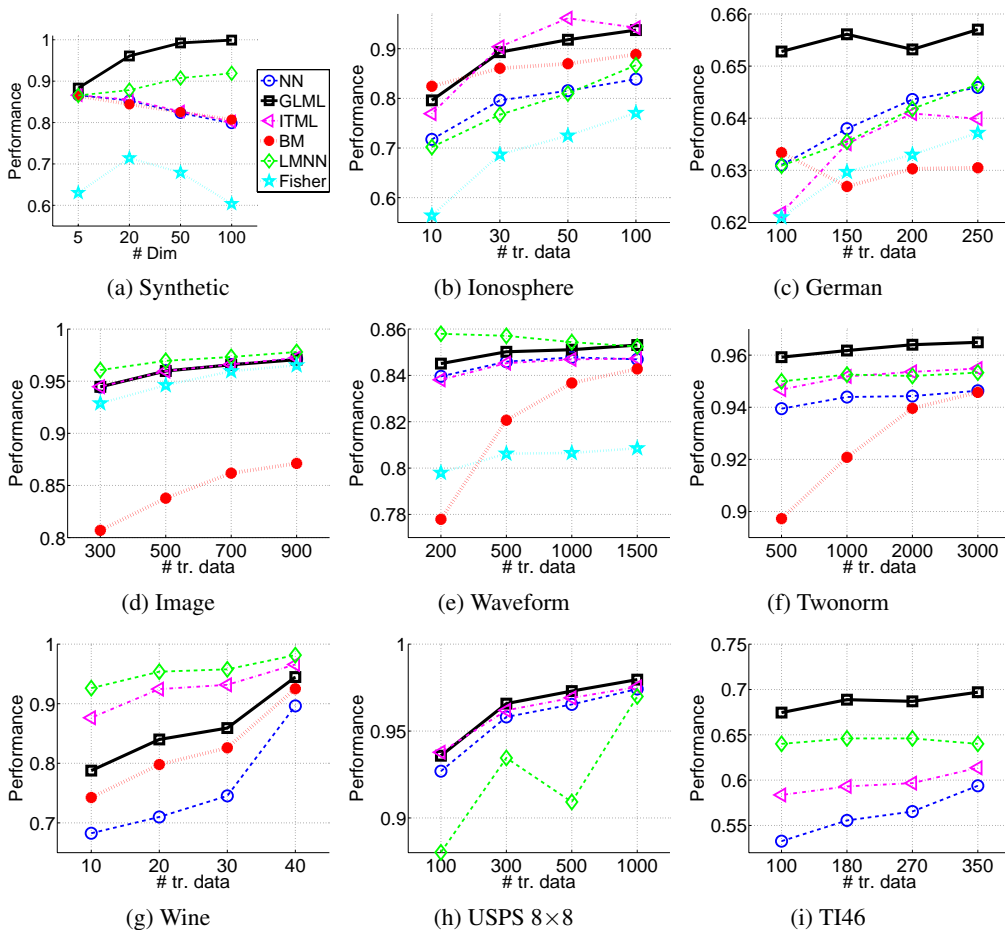

Figure 3: (a) Gaussian synthetic data with different dimensionality. As number of dimensions gets large, most methods degrade except GLML and LMNN. GLML continues to improve vastly over other methods. (b)∼(h) are the experiments on benchmark datasets varying the number of training data per class. (i) TI46 is the speech dataset pronounced by 8 men and 8 women. The Fisher kernel and BM are omitted for (f)∼(i) and (h)∼(i) respectively, since their performances are much worse than the naive nearest neighbor classifier.

in machine learning. Here we used simple Gaussians for the generative models, but this could be also easily extended to include other possibilities such as mixture models, hidden Markov models, or other dynamic generative models.

The kernelization of this work is straightforward, and the extension to the k-nearest neighbor setting using the theoretical distribution of k-th nearest neighbors is an interesting future direction. Another possible future avenue of work is to combine dimensionality reduction and metric learning using this framework.

### Acknowledgments

This research was supported by National Research Foundation of Korea (2010-0017734, 2010-0018950, 314-2008-1-D00377) and by the MARS (KI002138) and BK-IT Projects.

## Footnotes

[1] http://userweb.cs.utexas.edu/ pjain/itml/

[2] http://code.google.com/p/boosting/

[3] http://www.cse.wustl.edu/ kilian/Downloads/LMNN.html

[4] http://archive.ics.uci.edu/ml/

[5] http://www.fml.tuebingen.mpg.de/Members/raetsch/benchmark

## References

[1] B. Alipanahi, M. Biggs, and A. Ghodsi. Distance metric learning vs. Fisher discriminant analysis. In *Proceedings of the 23rd national conference on Artificial intelligence*, pages 598–603, 2008.

[2] K. Das and Z. Nenadic. Approximate information discriminant analysis: A computationally simple heteroscedastic feature extraction technique. *Pattern Recognition*, 41(5):1548–1557, 2008.

[3] J.V. Davis, B. Kulis, P. Jain, S. Sra, and I.S. Dhillon. Information-theoretic metric learning. In *Proceedings of the 24th International Conference on Machine Learning*, pages 209–216, 2007.

[4] R.O. Duda, P.E. Hart, and D.G. Stork. *Pattern Classification (2nd Edition)*. Wiley-Interscience, 2000.

[5] A. Frome, Y. Singer, and J. Malik. Image retrieval and classification using local distance functions. In *Advances in Neural Information Processing Systems 18*, pages 417–424, 2006.

[6] K. Fukunaga. *Introduction to Statistical Pattern Recognition*. Academic Press, San Diego, CA, 1990.

[7] K. Fukunaga and T.E. Flick. The optimal distance measure for nearest neighbour classification. *IEEE Transactions on Information Theory*, 27(5):622–627, 1981.

[8] K. Fukunaga and T.E. Flick. An optimal global nearest neighbour measure. *IEEE Transactions on Pattern Analysis and Machine Intelligence*, 6(3):314–318, 1984.

[9] A. Globerson and S. Roweis. Metric learning by collapsing classes. In *Advances in Neural Information Processing Systems 18*, pages 451–458. 2006.

[10] J. Goldberger, S. Roweis, G. Hinton, and R. Salakhutdinov. Neighbourhood components analysis. In *Advances in Neural Information Processing Systems 17*, pages 513–520. 2005.

[11] M. N. Goria, N. N. Leonenko, V. V. Mergel, and P. Inverardi. A new class of random vector entropy estimators and its applications in testing statistical hypotheses. *Journal of Nonparametric Statistics*, 17(3):277–297, 2005.

[12] T. Jaakkola and D. Haussler. Exploiting generative models in discriminative classifiers. In *Advances in Neural Information Processing Systems 11*, pages 487–493, 1998.

[13] J.N. Kapur. *Measures of Information and Their applications*. John Wiley & Sons, New York, NY, 1994.

[14] S. Lacoste-Julien, F. Sha, and M. Jordan. DiscLDA: Discriminative learning for dimensionality reduction and classification. In *Advances in Neural Information Processing Systems 21*, pages 897–904. 2009.

[15] J.A. Lasserre, C.M. Bishop, and T.P. Minka. Principled hybrids of generative and discriminative models. In *Proceedings of the 2006 IEEE Computer Society Conference on Computer Vision and Pattern Recognition*, pages 87–94, 2006.

[16] M. Loog and R.P.W. Duin. Linear dimensionality reduction via a heteroscedastic extension of LDA: The chernoff criterion. *IEEE Transactions on Pattern Analysis and Machine Intelligence*, 26(6):732–739, 2004.

[17] Z. Nenadic. Information discriminant analysis: Feature extraction with an information-theoretic objective. *IEEE Transactions on Pattern Analysis and Machine Intelligence*, 29(8):1394–1407, 2007.

[18] A.Y. Ng and M.I. Jordan. On discriminative vs. generative classifiers: A comparison of logistic regression and naive Bayes. In *Advances in Neural Information Processing Systems 14*, pages 841–848, 2001.

[19] F. Perez-Cruz. Estimation of information theoretic measures for continuous random variables. In *Advances in Neural Information Processing Systems 21*, pages 1257–1264. 2009.

[20] R. Raina, Y. Shen, A.Y. Ng, and A. McCallum. Classification with hybrid generative/discriminative models. In *Advances in Neural Information Processing Systems 16*, pages 545–552. 2004.

[21] C. Shen, J. Kim, L. Wang, and A. van den Hengel. Positive semidefinite metric learning with boosting. In *Advances in Neural Information Processing Systems 22*, pages 1651–1659. 2009.

[22] N. Singh-Miller and M. Collins. Learning label embeddings for nearest-neighbor multi-class classification with an application to speech recognition. In *Advances in Neural Information Processing Systems 22*, pages 1678–1686. 2009.

[23] D. Tran and A. Sorokin. Human activity recognition with metric learning. In *Proceedings of the 10th European Conference on Computer Vision*, pages 548–561, 2008.

[24] Q. Wang, S. R. Kulkarni, and S. Verdú. A nearest-neighbor approach to estimating divergence between continuous random vectors. In *Proceedings of IEEE International Symposium on Information Theory*, pages 242–246, 2006.

[25] Q. Wang, S. R. Kulkarni, and S. Verdú. Divergence estimation for multidimensional densities via k-nearest-neighbor distances. *IEEE Transactions on Information Theory*, 55(5):2392–2405, 2009.

[26] K. Weinberger, J. Blitzer, and L. Saul. Distance metric learning for large margin nearest neighbor classification. In *Advances in Neural Information Processing Systems 18*, pages 1473–1480. 2006.

